# Generalized² Linear² Models

**Geoffrey J. Gordon**
*ggordon@cs.cmu.edu*

## Abstract

We introduce the Generalized² Linear² Model, a statistical estimator which combines features of nonlinear regression and factor analysis. A $(GL)^2M$ approximately decomposes a rectangular matrix $X$ into a simpler representation $f(g(A)h(B))$. Here $A$ and $B$ are low-rank matrices, while $f$, $g$, and $h$ are link functions. $(GL)^2Ms$ include many useful models as special cases, including principal components analysis, exponential-family PCA, the infomax formulation of independent components analysis, linear regression, and generalized linear models. They also include new and interesting special cases, one of which we describe below. We also present an iterative procedure which optimizes the parameters of a $(GL)^2M$. This procedure reduces to well-known algorithms for some of the special cases listed above; for other special cases, it is new.

## 1 Introduction

Let the $m \times n$ matrix $X$ contain an independent sample from some unknown distribution. Each column of $X$ represents a training example, and each row represents a measured feature of the examples. It is often reasonable to assume that some of the features are redundant, that is, that there exists a reduced set of $l$ features which contains all or most of the information in $X$.

If the reduced features are linear functions of the original features and the distributions of the elements of $X$ are Gaussian, redundancy means we can write $X$ as the product of two smaller matrices $U$ and $V$ with small sum of squared errors. This factorization is essentially a singular value decomposition: $U$ must span the first $l$ dimensions of the left principal subspace of $X$, while $V^\mathrm{T}$ must span the first $l$ dimensions of the right principal subspace. (Since the above requirements do not uniquely determine $U$ and $V$, the SVD traditionally imposes additional restrictions which we will ignore in this paper.)

The SVD has a long list of successes in machine learning, including information retrieval applications such as latent semantic analysis [1] and link analysis [2]; pattern recognition applications such as "eigenfaces" [3]; structure from motion algorithms [4]; and data compression tools [5]. Unfortunately, the SVD makes two assumptions which can limit its accuracy as a learning tool.

The first assumption is the use of the sum of squared errors between $X$ and $UV$ as a loss function. Squared error loss means that predicting 1000 when the answer is 1010 is as bad as saying -7 when the answer is 3. The second assumption is that

the reduced features are linear functions of the original features. Instead, $X$ might be a nonlinear function of $UV$, and $U$ and $V$ might be nonlinear functions of some other matrices $A$ and $B$. To address these shortcomings, we propose the model

$$\bar{X} = f(g(A)h(B)) \qquad (1)$$

for the expected value of $X$. We also propose allowing non-quadratic loss functions for the error $(X - \bar{X})$ and the parameter matrices $A$ and $B$. The fixed functions

$$f : \mathbb{R}^{m \times n} \mapsto \mathbb{R}^{m \times n} \qquad g : \mathbb{R}^{m \times l} \mapsto \mathbb{R}^{m \times l} \qquad h : \mathbb{R}^{l \times n} \mapsto \mathbb{R}^{l \times n}$$

are called *link functions*. By analogy to generalized linear models [6], we call equation (1) a Generalized$^2$ Linear$^2$ Model: generalized$^2$ because it uses link functions for the parameters $A$ and $B$ as well as the prediction $\bar{X}$, and linear$^2$ because like the SVD it is bilinear.

As long as we choose link and loss functions that match each other (see below for the definition of matching link and loss), there will exist efficient algorithms for finding $A$ and $B$ given $X$, $f$, $g$, and $h$. Because (1) is a generalization of the SVD, (GL)$^2$Ms are drop-in replacements for SVDs in all of the applications mentioned above, with better reconstruction performance when the SVD's error model is inaccurate. In addition, they open up new applications (see section 7 for one) where an SVD would have been unable to provide a sufficiently accurate reconstruction.

## 2   Matching link and loss functions

Whenever we try to optimize the predictions of a nonlinear model, we need to worry about getting stuck in local minima. One example of this problem is when we try to fit a single sigmoid unit with parameters $\theta \in \mathbb{R}^d$ to training inputs $x_i \in \mathbb{R}^d$ and target outputs $y_i \in \mathbb{R}$ under squared error loss:

$$L = \sum_i (y_i - \hat{y}_i)^2 \qquad \hat{y}_i = \mathrm{logit}(z_i) \qquad z_i = x_i \cdot \theta \qquad \mathrm{logit}(z) = (1 + e^{-z})^{-1}$$

Even for small training sets, the number of local minima of $L$ can grow exponentially with the dimension $d$ [7]. On the other hand, if we optimize the same predictions $\hat{y}_i$ under the logarithmic loss function $\sum_i [y_i \log \hat{y}_i + (1 - y_i) \log(1 - \hat{y}_i)]$ instead of squared error, our optimization problem is convex. Because the logistic link works with the log loss to produce a convex optimization problem, we say they *match* each other [7]. Matching link-loss pairs are important because minimizing a convex loss function is usually far easier than minimizing a nonconvex one.

We can use any convex function $F(z)$ to generate a matching pair of link and loss functions. The loss function which corresponds to $F$ is

$$D_F(z \mid y) \equiv \sum_i [F(z_i) - y_i z_i + F^*(y_i)] \qquad (2)$$

where $F^*(y)$ is defined so that $\min_z D_F(z \mid y) = 0$. ($F^*$ is the *convex dual* of $F$ [8], and $D_F$ is the *generalized Bregman divergence* from $z$ to $y$ [9].)

Expression (2) is nonnegative, and it is globally convex in all of the $z_i$s (and therefore also in $\theta$ since each $z_i$ is a linear function of $\theta$). If we write $f$ for the gradient of $F$, the derivative of (2) with respect to $z_i$ is $f(z_i) - y_i$. So, (2) will be zero if and only if $y_i = f(z_i)$ for all $i$; in other words, using the loss (2) implies that $\hat{y}_i = f(z_i)$ is our best prediction of $y_i$, and $f$ is therefore our matching link function.

We will need two facts about convex duals below. The first is that $F^*$ is always convex, and the second is that the gradient of $F^*$ is equal to $f^{-1}$. (Also, convex duality is defined even when $F$, $G$, and $H$ aren't differentiable. If they are not, replace derivatives by subgradients below.)

## 3 Loss functions for (GL)²Ms

In (GL)²Ms, matching loss functions will be particularly important because we need to deal with three separate nonlinear link functions. We will usually not be able to avoid local minima entirely; instead, the overall loss function will be convex in some groups of parameters if we hold the remaining parameters fixed.

We will specify a (GL)²M by picking three link functions and their matching loss functions. We can then combine these individual loss functions into an overall loss function as described in section 4; fitting a (GL)²M will then reduce to minimizing the overall loss function with respect to our parameters. Each choice of links results in a different (GL)²M and therefore potentially a different decomposition of $X$.

The choice of link functions is where we should inject our domain knowledge about what sort of noise there is in $X$ and what parameter matrices $A$ and $B$ are a priori most likely. Useful link functions include $f(x) = x$ (corresponding to squared error and Gaussian noise), $f(x) = \log x$ (unnormalized KL-divergence and Poisson noise), and $f(x) = (1 + e^{-x})^{-1}$ (log-loss and Bernoulli noise).

The loss functions themselves are only necessary for the analysis; all of our algorithms need only the link functions and (in some cases) their derivatives. So, we can pick the loss functions and differentiate to get the matching link functions; or, we can pick the link functions directly and not worry about the corresponding loss functions. In order for our analysis to apply, the link functions must be derivatives of some (possibly unknown) convex functions.

Our loss functions are $D_F$, $D_G$, and $D_H$ where

$$F : \mathbb{R}^{m \times n} \mapsto \mathbb{R} \qquad G : \mathbb{R}^{m \times l} \mapsto \mathbb{R} \qquad H : \mathbb{R}^{l \times n} \mapsto \mathbb{R}$$

are convex functions. We will abuse notation and call $F$, $G$, and $H$ loss functions as well: $F$ is the prediction loss, and its derivative $f$ is the prediction link; it provides our model of the noise in $X$. $G$ and $H$ are the parameter losses, and their derivatives $g$ and $h$ are the parameter links; they tell us which values of $A$ and $B$ are a priori most likely. By convention, since $F$ takes an $m \times n$ matrix argument, we will say that the input and output to $f$ are also $m \times n$ matrices (similarly for $g$ and $h$).

## 4 The model and its fixed point equations

We will define a (GL)²M by specifying an overall loss function which relates the parameter matrices $A$ and $B$ to the data matrix $X$. If we write $U = g(A)$ and $V = h(B)$, the (GL)²M loss function is

$$L(U, V) = F(UV) - X \circ UV + G^*(U) + H^*(V) \tag{3}$$

Here $A \circ B$ is the "matrix dot product," often written $\mathrm{tr}(A^\mathrm{T} B)$.

Expression (3) is a sum of three Bregman divergences, ignoring terms which don't depend on $U$ and $V$: it is $D_F(UV \mid X) + D_G(0 \mid U) + D_H(0 \mid V)$. The $F$-divergence tends to pull $UV$ towards $X$, while the $G$- and $H$-divergences favor small $U$ and $V$.

To further justify (3), we can examine what happens when we compute its derivatives with respect to $U$ and $V$ and set them to 0. The result is a set of fixed-point equations that the optimal parameter settings must satisfy:

$$U^\mathrm{T}(X - f(UV)) = B \tag{4}$$
$$(X - f(UV))V^\mathrm{T} = A \tag{5}$$

To understand these equations, we can consider two special cases. First, if we let $G^*$ go to zero (so there is no pressure to keep $U$ and $V$ small), (4) becomes

$$U^{\mathrm{T}}(X - f(UV)) = 0 \qquad (6)$$

which tells us that each column of the error matrix must be orthogonal to each column of $U$. Second, if we set the prediction link to be $f(UV) = UV$, (6) becomes

$$U^{\mathrm{T}}UV = U^{\mathrm{T}}X$$

which tells us that for a given $U$, we must choose $V$ so that $UV$ reconstructs $X$ with the smallest possible sum of squared errors.

## 5  Algorithms for fitting (GL)²Ms

We could solve equations (4–5) with any of several different algorithms. For example, we could use gradient descent on either $U, V$ or $A, B$. Or, we could use the generalized gradient descent [9] update rule (with learning rate $\alpha$):

$$A \leftarrow_\alpha (X - f(UV))V^{\mathrm{T}} \qquad B \leftarrow_\alpha U^{\mathrm{T}}(X - f(UV))$$

The advantage of these algorithms is that they are simple to implement and don't require additional assumptions on $F$, $G$, and $H$. They can even work when $F$, $G$, and $H$ are nondifferentiable by using subgradients.

In this paper, though, we will focus on a different algorithm. Our algorithm is based on Newton's method, and it reduces to well-known algorithms for several special cases of (GL)²Ms. Of course, since the end goal is solving (4–5), this algorithm will not always be the method of choice; instead, any given implementation of a (GL)²M should use the simplest algorithm that works.

For our Newton algorithm we will need to place some restrictions on the prediction and parameter loss functions. (These restrictions are only necessary for the Newton algorithm; more general loss functions still give valid (GL)²Ms, but require different algorithms.) First, we will require (4–5) to be differentiable. Second, we will restrict

$$F(Z) = \sum_{ij} F_{ij}(Z_{ij}) \qquad G(A) = \sum_i G_i(A_{i.}) \qquad H(B) = \sum_j H_j(B_{.j})$$

These definitions fix most of the second derivatives of $L(U, V)$ to be zero, simplifying and speeding up computation. Write $f_{ij}$, $g_i$, and $h_j$ for the respective derivatives.

With these restrictions, we can linearize (4) and (5) around our current guess at the parameters, then solve the resulting equations to find updated parameters. To simplify notation, we can think of (4) as $j$ separate equations, one for each column of $V$. Linearizing with respect to $V_{.j}$ gives:

$$(U^{\mathrm{T}}D_j U + \mathbf{H}_j)(V_{.j}^{\mathrm{new}} - V_{.j}) = U^{\mathrm{T}}(X_{.j} - f_{.j}(UV_{.j})) - B_{.j}$$

where the $l \times l$ matrix $\mathbf{H}_j$ is the Hessian of $H_j^*$ at $V_{.j}$, or equivalently the inverse of the Hessian of $H_j$ at $B_{.j}$; and the $m \times m$ diagonal matrix $D_j$ contains the second derivatives of $F$ with respect to the $j$th column of its argument. That is,

$$\mathbf{H}_j = \left(\tfrac{\mathrm{d}}{\mathrm{d}V_{.j}}\right)^2 H_j^*(V_{.j}) = [h_j'(B_{.j})]^{-1} \qquad D_j = \mathrm{diag}(f_{.j}'(UV_{.j}))$$

Now, collecting terms involving $V_{.j}^{\mathrm{new}}$ yields:

$$(U^{\mathrm{T}}D_j U + \mathbf{H}_j)V_{.j}^{\mathrm{new}} = U^{\mathrm{T}}D_j(UV_{.j} + D_j^{-1}(X_{.j} - f_{.j}(UV_{.j}))) + \mathbf{H}_j V_{.j} - B_{.j} \quad (7)$$

We can recognize (7) as a weighted least squares problem with weights $\sqrt{D_j}$, prior precision $\mathbf{H}_j$, prior mean $V_{\cdot j} + \mathbf{H}_j^{-1} B_{\cdot j}$, and target outputs

$$UV_{\cdot j} + D_j^{-1}(X_{\cdot j} - f(UV_{\cdot j}))$$

Similarly, we can linearize with respect to rows of $U$ to find the equation

$$U_{i\cdot}^{\text{new}}(VD_iV^{\mathrm{T}} + \mathbf{G}_i) = ((X_{i\cdot} - f_{i\cdot}(U_{i\cdot}V))D_i^{-1} + U_{i\cdot}V)D_iV^{\mathrm{T}} + U_{i\cdot}\mathbf{G}_i - A_{i\cdot}. \quad (8)$$

where $\mathbf{G}_i$ is the Hessian of $G_i^*$ and $D_i$ contains the second derivatives of $F$ with respect to the $i$th row of its argument.

We can solve one copy of (7) simultaneously for each column of $V$, then replace $V$ by $V^{\text{new}}$. Next we can solve one copy of (8) simultaneously for each row of $U$, then replace $U$ by $U^{\text{new}}$. Alternating between these two updates will tend to reduce (3).[1]

# 6 Related models

There are many important special cases of (GL)$^2$Ms. We derive two in this section; others include principal components analysis, "sensible" PCA, linear regression, generalized linear models, and the weighted majority algorithm. (Our Newton algorithm turns into power iteration for PCA and iteratively-reweighted least squares for GLMs.) (GL)$^2$Ms are related to generalized bilinear models; the latter include some of the above special cases, but not ICA, weighted majority, or the example of section 7. There are natural generalizations of (GL)$^2$Ms to multilinear interactions. Finally, some models such as non-negative matrix factorization [10] and generalized low-rank approximation [11] are cousins of (GL)$^2$Ms: they use a loss function which is convex in either factor with the other fixed but which is not a Bregman divergence.

## 6.1 Independent components analysis

In ICA, we assume that there is a hidden matrix $V$ (the same size as $X$) of independent random variables, and that $X$ was generated from $V$ by applying a square matrix $U$. We seek to recover the mixing matrix $U$ and the sources $V$; in other words, we want to decompose $X = UV$ so that the elements of $V$ are as nearly independent as possible.

The infomax algorithm for ICA assumes that the elements of $V$ follow distributions with heavy tails (i.e., high kurtosis). This assumption helps us find independent components because the sum of two heavy-tailed random variables tends to have lighter tails, so we can search for $U$ by trying to make the elements of $V$ follow a heavy-tailed distribution.

In our notation, the fixed point of the infomax algorithm for ICA is

$$-U^{\mathrm{T}} = \tanh(V)X^{\mathrm{T}} \quad (9)$$

(see, e.g., equation (11) or (13) of [12]). To reproduce (9), we will let the prediction link $f$ be the identity, and we will let the duals of the parameter loss functions be

$$G^*(U) = -\epsilon \log \det U$$
$$H^*(V) = \epsilon \sum_{ij} \log \cosh v_{ij}$$

where $\epsilon$ is a small positive real number. Then equations (4) and (5) become

$$U^{\mathrm{T}}(X - UV) = \epsilon \tanh(V) \qquad (10)$$
$$(X - UV)V^{\mathrm{T}} = -\epsilon U^{-\mathrm{T}} \qquad (11)$$

since the derivative of $\log \cosh v$ is $\tanh v$ and the derivative of $\log \det U$ is $U^{-\mathrm{T}}$.

Right-multiplying (10) by $(UV)^{\mathrm{T}}$ and substituting in (11) yields

$$-U^{\mathrm{T}} = \tanh(V)(UV)^{\mathrm{T}} \qquad (12)$$

Now since $UV \to X$ as $\epsilon \to 0$, (12) is equivalent to (9) in the limit of vanishing $\epsilon$.

## 6.2 Exponential family PCA

To duplicate exponential family PCA [13], we can set the prediction link $f$ arbitrarily and let the parameter links $g$ and $h$ be large multiples of the identity. Our Newton algorithm is applicable under the assumptions of [13], and (7) becomes

$$U^{\mathrm{T}} D_j U V_{\cdot j}^{\mathrm{new}} = U^{\mathrm{T}} D_j (UV_{\cdot j} + D_j^{-1}(X_{\cdot j} - f_{\cdot j}(UV_{\cdot j}))) \qquad (13)$$

Equation (13) along with the corresponding modification of (8) should provide a much faster algorithm than the one proposed in [13], which updates only part of $U$ or $V$ at a time and keeps updating the same part until convergence before moving on to the next one.

# 7   Example: robot belief states

Figure 1 shows a map of a corridor in the CMU CS building. A robot navigating in this corridor can sense both side walls and compute an accurate estimate of its lateral position. Unless it is near an observable feature such the lab door near the middle of the corridor, however, it can't accurately resolve its position along the corridor and it can't tell whether it is pointing left or right.

In order to plan to achieve a goal in this environment, the robot must maintain a belief state (a probability distribution representing its best information about the unobserved state variables). The map shows the robot's starting belief state: it is at one end of the corridor facing in, but it doesn't know which end. We collected a training set of 400 belief states by driving the robot along the corridor and feeding its sensor readings to a belief tracker [14]. To simulate a larger environment with greater uncertainty, we artificially reduced sensor range and increased error. Figure 1 shows two of the collected beliefs.

Planning is difficult because belief states are high-dimensional: even in this simple world there are 550 states (275 positions at 10cm intervals along the corridor $\times$ 2 orientations), so a belief is a vector in $\mathbb{R}^{550}$. Fortunately, the robot never encounters most belief states. This regularity can make planning tractable: if we can identify a few features which extract the important information from belief states, we can plan in low-dimensional feature space instead of high-dimensional belief space.

We factored the matrix of belief states using feature space ranks $l = 3, 4, 5$. For the prediction link $f(Z)$ we used $\exp(Z)$ (componentwise); this link ensures that the predicted beliefs are positive, and treats errors in small probabilities as proportionally more important than errors in large ones. (The matching loss for $f$ is a Poisson log-likelihood or unnormalized KL-divergence.) For the parameter link $h$ we used $10^{12}I$, corresponding to $H^* = 10^{-12}\|V\|^2/2$ (a weak bias towards small $V$).

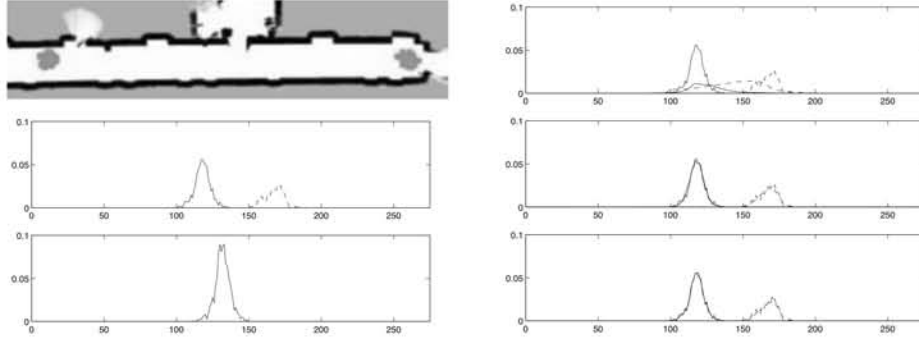

Figure 1: Belief states. Left panel: overhead map of corridor with initial belief $b_1$; belief state $b_{80}$ (just before robot finds out which direction it's pointing); belief $b_{90}$ (just after finding out). Right panel: reconstruction of $b_{80}$ with 3, 4, and 5 features.

We set $G^* = 10^{-12}\|U\|^2/2 + \Delta(U)$, where $\Delta$ is 0 when the first column of $U$ contains all 1s and $\infty$ otherwise. This loss function fixes the first column of $U$, representing our knowledge that one feature should be a normalizing constant so that each belief sums to 1. The subgradient of $G^*$ is $10^{-12}U + [k, 0]$, so equation (5) becomes

$$(X - \exp(UV))V^{\mathrm{T}} = 10^{-12}U + [k, 0]$$

Here $[k, 0]$ is a matrix with an arbitrary first column and all other elements 0; this matrix has enough degrees of freedom to compensate for the constraints on $U$.

Our Newton algorithm handles this modified fixed point equation without difficulty. So, this $(\mathrm{GL})^2\mathrm{M}$ is a principled and efficient way to decompose a matrix of probability distributions. So far as we know this model and algorithm have not been described in the literature.

Figure 1 shows our reconstructions of a representative belief state using $l = 3, 4, 5$ features (one of which is a normalizing constant that can be discarded for planning). The $l = 5$ reconstruction is consistently good across all 400 beliefs, while the $l = 4$ reconstruction has minor artifacts for some beliefs. A small number of restarts is required to achieve good decompositions for $l = 3$ where the optimization problem is most constrained. For comparison, a traditional SVD requires a matrix of rank about 25 to achieve the same mean-squared reconstruction error as our rank-3 decomposition. It requires rank about 85 to match our rank-5 decomposition.

Examination of the learned $U$ matrix (not shown) for $l = 4$ reveals that the corridor is mapped into two smooth curves in feature space, one for each orientation. Corresponding states with opposite orientations are mapped into similar feature vectors for one half of the corridor (where the training beliefs were sometimes confused about orientation) but not the other (where there were no training beliefs that indicated any connection between orientations). Reconstruction artifacts occur when a curve nearly self-intersects and causes confusion between states. This self-intersection happens because of local minima in the loss function; with more flexibility ($l = 5$) the optimizer is able to untangle the curves and avoid self-intersection.

Our success in compressing the belief state translates directly into success in planning; see [15] for details. By comparison, traditional SVD on either the beliefs or the log beliefs produces feature sets which are unusable for planning because they do not achieve sufficiently good reconstruction with few enough features.

# 8  Discussion

We have introduced a new general class of nonlinear regression and factor analysis model, presenting a derivation and algorithm, reductions to previously-known special cases, and a practical example. The model is a drop-in replacement for PCA, but can provide much better reconstruction performance in cases where the PCA error model is inaccurate. Future research includes online algorithms for parameter adjustment; extensions for missing data; and exploration of new link functions.

## Acknowledgments

Thanks to Nick Roy for helpful comments and for providing the data analyzed in section 7. This work was supported by AFRL contract F30602–01–C–0219, DARPA's MICA program, and by AFRL contract F30602–98–2–0137, DARPA's CoABS program. The opinions and conclusions are the author's and do not reflect those of the US government or its agencies.

## Footnotes

[1]To guarantee convergence, we can check that (3) decreases and reduce our step size if we encounter problems. (Since $U^{\mathrm{T}}D_jU$, $\mathbf{H}_j$, $VD_iV^{\mathrm{T}}$, and $\mathbf{G}_i$ are all positive definite, the Newton update directions are descent directions; so, there always exists a small enough step size.) We have not found this check necessary in practice.

# References

[1] T. K. Landauer, P. W. Foltz, and D. Laham. Introduction to latent semantic analysis. *Discourse Processes*, 25:259–284, 1998.

[2] Jon M. Kleinberg. Authoritative sources in a hyperlinked environment. *Journal of the ACM*, 46(5):604–632, 1999.

[3] M. Turk and A. Pentland. Eigenfaces for recognition. *Journal of Cognitive Neuroscience*, 3(1):71–86, 1991.

[4] Carlo Tomasi and Takeo Kanade. Shape and motion from image streams under orthography: a factorization method. *Int. J. Computer Vision*, 9(2):137–154, 1992.

[5] D. P. O'Leary and S. Peleg. Digital image compression by outer product expansion. *IEEE Trans. Communications*, 31:441–444, 1983.

[6] P. McCullagh and J. A. Nelder. *Generalized Linear Models*. Chapman & Hall, London, 2nd edition, 1983.

[7] Peter Auer, Mark Hebster, and Manfred K. Warmuth. Exponentially many local minima for single neurons. In *NIPS*, vol. 8. MIT Press, 1996.

[8] R. Tyrell Rockafellar. *Convex Analysis*. Princeton University Press, New Jersey, 1970.

[9] Geoffrey J. Gordon. *Approximate Solutions to Markov Decision Processes*. PhD thesis, Carnegie Mellon University, 1999.

[10] Daniel Lee and H. Sebastian Seung. Algorithms for nonnegative matrix factorization. In *NIPS*, vol. 13. MIT Press, 2001.

[11] Nathan Srebro. Personal communication, 2002.

[12] Anthony J. Bell and Terrence J. Sejnowski. The 'independent components' of natural scenes are edge filters. *Vision Research*, 37(23):3327–3338, 1997.

[13] Michael Collins, Sanjoy Dasgupta, and Robert Schapire. A generalization of principal component analysis to the exponential family. In *NIPS*, vol. 14. MIT Press, 2002.

[14] D. Fox, W. Burgard, F. Dellaert, and S. Thrun. Monte Carlo localization: Efficient position estimation for mobile robots. In *AAAI*, 1999.

[15] Nicholas Roy and Geoffrey J. Gordon. Exponential family PCA for belief compression in POMDPs. In *NIPS*, vol. 15. MIT Press, 2003.

[16] Sam Roweis. EM algorithms for PCA and SPCA. In *NIPS*, vol. 10. MIT Press, 1998.
